# TRAFFIC: Recognizing Objects Using Hierarchical Reference Frame Transformations

**Richard S. Zemel**
Computer Science Dept.
University of Toronto
Toronto, ONT M5S 1A4

**Michael C. Mozer**
Computer Science Dept.
University of Colorado
Boulder, CO 80309-0430

**Geoffrey E. Hinton**
Computer Science Dept.
University of Toronto
Toronto, ONT M5S 1A4

## ABSTRACT

We describe a model that can recognize two-dimensional shapes in an unsegmented image, independent of their orientation, position, and scale. The model, called TRAFFIC, efficiently represents the structural relation between an object and each of its component features by encoding the fixed viewpoint-invariant transformation from the feature's reference frame to the object's in the weights of a connectionist network. Using a hierarchy of such transformations, with increasing complexity of features at each successive layer, the network can recognize multiple objects in parallel. An implementation of TRAFFIC is described, along with experimental results demonstrating the network's ability to recognize constellations of stars in a viewpoint-invariant manner.

## 1   INTRODUCTION

A key goal of machine vision is to recognize familiar objects in an unsegmented image, independent of their orientation, position, and scale. Massively parallel models have long been used for lower-level vision tasks, such as primitive feature extraction and stereo depth. Models addressing "higher-level" vision have generally been restricted to pattern matching types of problems, in which much of the inherent complexity of the domain has been eliminated or ignored.

The complexity of object recognition stems primarily from the difficult search required to find the correspondence between features of candidate objects and image

features. Images contain spurious features, which do not correspond to any object features; objects in an image may have missing or occluded features; and noisy measurements make it impossible to align object features to image features exactly. These problems are compounded in realistic domains, where images are not segmented and normalized and the number of candidate objects is large.

In this paper, we present a structured, general model of object recognition – called *TRAFFIC* (a loose acronym for "transforming feature instances") – that addresses these difficult problems through a combination of strategies. First, we directly build constraints on the spatial relationships between features of an object directly into the architecture of a connectionist network. We thereby limit the space of possible matches by constructing only plausible assignments of image features to objects. Second, we embed this construction into a hierarchical architecture, which allows the network to handle unsegmented, non-normalized images, and also allows for a wide range of candidate objects. Third, we allow TRAFFIC to discover the critical spatial relationships among features through training on examples of the target objects in various poses.

## 2    MODEL HIGHLIGHTS

The following sections outline the three fundamental aspects of TRAFFIC. For a more complete discussion of the details of TRAFFIC, see (Zemel, 1989).

### 2.1    ENCODING STRUCTURAL RELATIONS

The first key aspect of TRAFFIC concerns its encoding and use of the fixed spatial relations between a rigid object and each of its component features. If we assume that each feature has an intrinsic reference frame, then for a rigid object and a particular feature of that object, there is a *fixed* viewpoint-independent transformation from the feature's reference frame to the object's. This transformation can be used to predict the object's reference frame from the feature's. To recognize objects, TRAFFIC takes advantage of the fact that all features of the same object will predict the identical reference frame for that object (the "viewpoint consistency constraint" (Lowe, 1987)).

Each reference frame transformation can be expressed as a matrix multiplication that is efficiently implemented in a connectionist network. Consider a two-layer network, with one layer containing units representing particular features, the other containing units representing objects. For two-dimensional shapes, each feature is described by a set of four *instantiation units*. These real-valued units represent the parameter values associated with the feature: $(x,y)$-position, orientation, and scale. The objects have a set of instantiation units as well. The units representing particular features are connected to the units representing each object containing that feature, thereby assigning each feature-object pair its own set of weighted connections. The fixed matrix that describes the transformation from the feature's intrinsic reference frame to the object's can be directly implemented in the set of weights connecting the instantiation units of the feature and the object.

We can describe any instantiation, or any transformation between instantiations, as a vector of four parameters. Let $P_{if} = (x_{if}, y_{if}, c_{if}, s_{if})$ specify the reference frame of the feature with respect to the image, where $x_{if}$ and $y_{if}$ represent the coordinates of the feature origin relative to the image frame, $c_{if}$ and $s_{if}$ represent the scale and angle of the feature frame w.r.t. the image frame. Rather than encoding these values directly, $c_{if}$ represents the product of the scale and the cosine of the angle, while $s_{if}$ represesents the product of the scale and the sine of the angle.[1] Let $P_{io} = (x_{io}, y_{io}, c_{io}, s_{io})$, specify the reference frame of the object with respect to the image. Finally, let $P_{fo} = (x_{fo}, y_{fo}, c_{fo}, s_{fo})$ specify the transformation from the reference frame of the object to that of the feature.

Each of these sets of parameters can be placed into a transformation matrix which converts points in one reference frame to points in another. We can express $P_{if}$ as the matrix $T_{if}$, a transformation from the feature frame to the image frame:

$$ T_{if} = \begin{bmatrix} c_{if} & s_{if} & x_{if} \\ -s_{if} & c_{if} & y_{if} \\ 0 & 0 & 1 \end{bmatrix} $$

Likewise, we can express $P_{fo}$ as the matrix $T_{fo}$, a transformation from the object to feature frame, and $P_{io}$ as $T_{io}$, a transformation from the object to image frame. Because $T_{fo}$ is fixed for a given feature-object pair and $T_{if}$ is derived from the image, $T_{io}$ can easily be computed by composing these two transforms: $T_{io} = T_{if}T_{fo}$.

The four parameters underlying $T_{io}$ can then be extracted, which results in the following four equations for $P_{io}$:

$$ \begin{aligned} x_{io} &= c_{if}x_{fo} + s_{if}y_{fo} + x_{if} \\ y_{io} &= -s_{if}x_{fo} + c_{if}y_{fo} + y_{if} \\ c_{io} &= c_{if}c_{fo} - s_{if}s_{fo} \\ s_{io} &= c_{if}s_{fo} + s_{if}c_{fo} \end{aligned} $$

This transformation is easily implemented in a network by connecting the units representing $P_{if}$ to the units representing $P_{io}$ with the appropriate weights (Figure 1). In this manner, TRAFFIC directly encodes the reference frame transformation from a feature to an object in the connections from the set of units representing the feature's reference frame to units representing the object's frame. The specification of an object's reference frame can therefore be derived directly from each of its component features on the basis of the structural relationship between the feature and the object. Because each feature of an object should predict the same reference frame parameters for the object, we can determine whether the object is really present in the image by checking to see if the various features make identical

[1] We represent angles by their sines and cosines to avoid the discontinuities involved in representing orientation by a single number and to eliminate the non-linear step of computing $\sin \theta_{if}$ from $\theta_{if}$. Note that we represent the four degrees of freedom in the instantiation parameters using four units; a neurally plausible extension to this scheme which does not require single units with arbitrary precision could allocate a *pool* of units to each of these parameters.

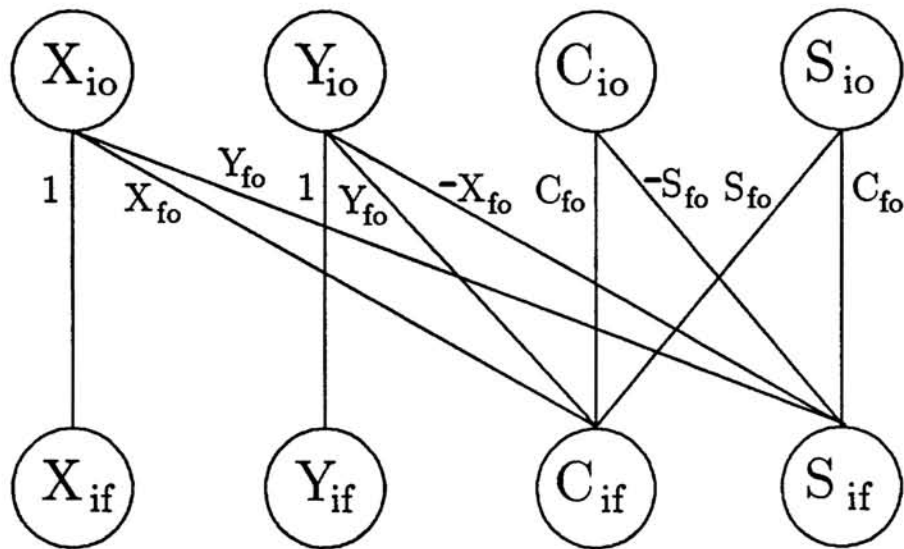

**Figure 1:** The matrix $T_{fo}$ is a fixed coordinate transformation from the reference frame of feature $f$ to the reference frame of object $o$. This figure shows how $T_{fo}$ can be built into the weights connecting the object-instantiation units and the feature-instantiation units.

predictions. In Section 2.3 we discuss how the object instantiation is formed in cases where the object parameters predicted by the features do not agree perfectly.

## 2.2    FEATURE ABSTRACTION HIERARCHY

TRAFFIC recursively extends the notion of reference frame transformations between features and objects in a hierarchical architecture. It is impractical to hope that any network will be able to directly map low-level input features to complex objects. The input features must be simple enough to be easily extracted from images without relying on sophisticated segmentation and interpretation. If they are simple, however, they will be unable to uniquely predict the object's reference frame, since a complex object may contain many copies of a single simple feature.

To address this problem, we adopt a hierarchical approach, introducing several layers of intermediate features between the input and output layers. In each layer, several features are grouped together to form an 'object' in the layer above; this 'object' then serves as a feature for 'objects' in the next layer. The lowest layer contains simple features, such as edges and various corner types. The objects to be recognized appear at the top of the hierarchy – the output layer of the network.

This composition hierarchy builds up a description of objects by selectively grouping sets of features, forming an increasingly abstract set of features. The power of this representation comes in the sharing of a set of features in one layer by objects in the layer above.

To represent multiple features of the same type simultaneously, we carve up the image into spatially-contiguous *regions*, each allowing the representation of one

instance of each feature. The network can thus represent several instances of a feature type simultaneously, provided they lie in different regions.

We tailor the regions to the abstraction hierarchy as follows. In the lowest layers, the features are simple and numerous, so we need many regions, but with only a few feature types per region. In upper layers of the hierarchy, the features become increasingly complex and span a larger area of the image; the number of feature types increases and the regions become larger, while the instantiation units retain accurate viewpoint information. In the highest layer, there is a single region, and it spans the entire original image. At this level, the network can recognize and specify parameters for a single instance of each object it has been trained on.

## 2.3   FORMING OBJECT HYPOTHESES

The third key aspect of TRAFFIC is its method of combining information from features to determine both an object's reference frame and an overall estimate of the likelihood that the object is actually present in the image. This likelihood, called the object's *confidence*, is represented by an additional unit associated with each object.

Each feature individually predicts the object's reference frame, and TRAFFIC forms a single vector of object instantiation-parameters by averaging the predicted instantiations, weighted by the confidence of their corresponding features.[2] Every set of units representing an object is sensitive to feature instances appearing in a fixed area of the image – the *receptive field* of the object. The confidence of the object is then a function of the confidence of the features lying in its receptive field, as well as the variance of their predictions, because low variance indicates a highly self-consistent object instantiation.

Once the network has been defined – the regions, receptive fields, and feature types specified at each level, and the reference frame transformations encoded in the weights – recognition occurs in a single bottom-up pass through the network. TRAFFIC accepts as input a set of simple features and a description of their pose in the image. At each layer in turn, the network forms many candidate object instantiations from the set of feature instantiations in the layer below, and then suppresses the object instantiations that are not consistently predicted by several of their component features. At the output level of the network, the confidence unit of each object describes the likelihood that that object is in the image, and its instantiation units specify its pose.

## 3   IMPLEMENTING TRAFFIC

The domain we selected for study involves the recognition of constellations of stars. This problem has several interesting properties:  the image is by nature unseg-

mented; there are many false partial matches; no bottom-up cues suggest a natural frame of reference; and it requires the ability to perform 2-D transformation-invariant recognition.

Each image contains the set of visible stars in a region of the sky. The input to TRAFFIC is a set of features that represent triples of stars in particular configurations. This input is computed by first dividing the image into regions and extracting every combination of three stars within each region. The star triplets (more precisely, the inner angles of the triangles formed by the triplets) are fed into an unsupervised competitive-learning network whose task is to categorize the configuration as one of a small number of types – the primitive feature types for the input layer of TRAFFIC.

The architecture we implemented had an input layer, two intermediate layers, and an output layer.[3] Eight constellations were to be recognized, each represented by a single unit in the output layer. We used a simple unsupervised learning scheme to determine the feature types in the intermediate layers of the hierarchy, working up sequentially from the input layer. During an initial phase of training, the system samples many regions of the sky at random, creating features at one layer corresponding to the frequently occurring combinations of features in the layer below. This scheme forms flexible intermediate representations tailored to the domain, but not hand-coded for the particular object set.

This sampling method determined the connection weights through the intermediate layers of the network. Back propagation was then used to set the weights between the penultimate layer and the output layer.[4] The entire network could have been trained using back propagation, but the combined unsupervised-supervised learning method we used is much simpler and quicker, and worked well for this problem.

## 4    EXPERIMENTAL RESULTS

We have run several experiments to test the main properties of the network, detailed further in (Zemel, 1989). Each image used in training and testing contained one of the eight target constellations, along with other nearby stars.

The first experiment tested the basic recognition capability of the system, as well as its ability to learn useful connections between objects and features. The training set consisted of a single view of each constellation. The second experiment examined the network's ability to recognize a constellation independent of its position and orientation in the image. We expanded the set of training images to include four different views of each of the eight constellations, in various positions and orientations. The test set contained two novel views of the eight constellations. In both experiments, the network quickly (< 150 epochs) learned to identify the target object. Learning was slower in the second experiment, but the network performance

was identical for the training and testing images.

The third experiment tested the network's ability not only to recognize an instance of a constellation, but to correctly specify its reference frame. In most simulations, the network produced a correct description of the target object instantiation across the training and testing images.

A final experiment confirmed that the network did *not* recognize an instance of an object when the features of the object were present in the input but were not in the correct relation to one another. The confidence level of the target object decreased proportionately as random noise was added to the instantiation parameters of input features. This shows that the upper layers of the network perform the important function of detecting the spatial relations of features from non-local areas of the image.

## 5  RELATED WORK

TRAFFIC resembles systems based on the Hough transform (Ballard, 1981; Hinton, 1981) in that evidence from various feature instances is combined using the viewpoint consistency constraint. However, while these Hough transform models need a unit for every possible viewpoint of an object, TRAFFIC reduces hardware requirements by using real-valued units to represent viewpoints.[5] TRAFFIC also resembles the approach of (Mjolsness, Gindi and Anandan, 1989), which relies on a large optimization search to simultaneously find the best set of object instantiations and viewpoint parameters to fit the image data. The TRAFFIC network carries out a similar type of search, but the limited connectivity and hierarchical architecture of the network constrains the search. The feature abstraction hierachy used in TRAFFIC is common to many recognition systems. The pattern recognition technique known as hierarchical synthesis (Barrow, Ambler and Burstall, 1972), employs a similar architecture, as do several connectionist models (Denker et al., 1989; Fukushima, 1980; Mozer, 1988). Each of these systems achieve position- and rotation-invariance by removing position information in the upper layers of the hierarchy. The TRAFFIC hierarchy, on the other hand, maintains and manipulates accurate viewpoint information throughout, allowing it to consider relations between features in non-local areas of the image.

## 6  CONCLUSIONS AND FUTURE WORK

The experiments demonstrate that TRAFFIC is capable of recognizing a limited set of two-dimensional objects in a viewpoint-independent manner based on the structural relations among components of the objects. We are currently testing the network's ability to perform multiple-object recognition and its robustness with respect to noise and occlusion. We are also currently developing a probabilistic framework for combining the various predictions to form the most likely object

instantiation hypothesis. This probabilistic framework may increase the robustness of the model and allow it to handle deviations from object rigidity.

Another extension to TRAFFIC we are currently exploring concerns the creation of a pre-processing network to specify reference frame information for input features directly from a raw image. We train this network using an unsupervised learning method based on the mutual information between neighboring image patches (Becker and Hinton, 1989). Our aim is to apply this method to learn the mappings from features to objects throughout the network hierarchy.

## Acknowledgements

This research was supported by grants from the Ontario Information Technology Research Center, grant 87-2-36 from the Alfred P. Sloan foundation, and a grant from the James S. McDonnell Foundation to Michael Mozer.

## Footnotes

[2]This averaging technique contains an implicit assumption that the maximum expected deviation of a prediction from the actual value is a function of the number of features, and that there will always be enough good values to smooth out any large deviations. We are currently exploring improved methods of forming object hypotheses.

[3]The details of the network, such as the number of regions and feature types per layer, the number of connections, etc., are discussed in (Zemel, 1989).

[4]In this implementation, we used a less efficient method of encoding the transformations than the method discussed in Section 2.1, but both versions perform the same transformations.

[5] Many other recognition systems, such as Lowe's SCERPO system (1985), represent object reference frame information as sets of explicit parameters.

## References

Ballard, D. H. (1981). Generalizing the Hough transform to detect arbitrary shapes. *Pattern Recognition*, 13(2):111–122.

Barrow, H. G., Ambler, A. P., and Burstall, R. M. (1972). Some techniques for recognising structures in pictures. In *Frontiers of Pattern Recognition*. Academic Press, New York, NY.

Becker, S. and Hinton, G. E. (1989). Spatial coherence as an internal teacher for a neural network. Technical Report Technical Report CRG-TR-89-7, University of Toronto.

Bolles, R. C. and Cain, R. A. (1982). Recognizing and locating partially visible objects: The local-feature-focus method. *International Journal of Robotics Research*, 1(3):57–82.

Denker, J. S., Gardner, W. L., Graf, H. P., Henderson, D., Howard, R. E., Hubbard, W., D., J. L., Baird, H. S., and Guyon, I. (1989). Neural network recognizer for hand-written zip code digits. In Touretzky, D. S., editor, *Advances in neural information processing systems I*, pages 323–331, San Mateo, CA. Morgan Kaufmann Publishers, Inc.

Fukushima, K. (1980). Neocognitron: A self-organizing neural network model for a mechanism of pattern recognition unaffected by shift in position. *Biological Cybernetics*, 36:193–202.

Hinton, G. E. (1981). A parallel computation that assigns canonical object-based frames of reference. In *Proceedings of the 7th International Joint Conference on Artificial Intelligence*, pages 683–685, Vancouver, BC, Canada.

Huttenlocher, D. P. and Ullman, S. (1987). Object recognition using alignment. In *First International Conference on Computer Vision*, pages 102–111, London, England.

Lowe, D. G. (1985). *Perceptual Organization and Visual Recognition*. Kluwer Academic Publishers, Boston.

Lowe, D. G. (1987). The viewpoint consistency constraint. *International Journal of Computer Vision*, 1:57–72.

Mjolsness, E., Gindi, G., and Anandan, P. (1989). Optimization in model matching and perceptual organization. *Neural Computation*, 1:218–299.

Mozer, M. C. (1988). The perception of multiple objects: A parallel, distributed processing approach. Technical Report 8803, University of California, San Diego, Institute for Cognitive Science.

Zemel, R. S. (1989). TRAFFIC: A connectionist model of object recognition. Technical Report Technical Report CRG-TR-89-2, University of Toronto.